# Hierarchical Topic Modeling for Analysis of Time-Evolving Personal Choices

**XianXing Zhang**
Duke University
xianxing.zhang@duke.edu

**David B. Dunson**
Duke University
dunson@stat.duke.edu

**Lawrence Carin**
Duke University
lcarin@ee.duke.edu

## Abstract

The nested Chinese restaurant process is extended to design a nonparametric topic-model tree for representation of human choices. Each tree path corresponds to a type of person, and each node (topic) has a corresponding probability vector over items that may be selected. The observed data are assumed to have associated temporal covariates (corresponding to the time at which choices are made), and we wish to impose that with increasing time it is more probable that topics deeper in the tree are utilized. This structure is imposed by developing a new "change point" stick-breaking model that is coupled with a Poisson and product-of-gammas construction. To share topics across the tree nodes, topic distributions are drawn from a Dirichlet process. As a demonstration of this concept, we analyze real data on course selections of undergraduate students at Duke University, with the goal of uncovering and concisely representing structure in the curriculum and in the characteristics of the student body.

## 1   Introduction

As time progresses, the choices humans make often change. For example, the types of products one purchases, as well as the types of people one befriends, often change or evolve with time. However, the choices one makes later in life are typically statistically related to choices made earlier. Such behavior is of interest when considering marketing to particular groups of people, at different stages of their lives, and it is also relevant for analysis of time-evolving social networks. In this paper we seek to develop a hierarchical tree structure for representation of this phenomena, with each tree path characteristic of a *type* of person, and a tree node defined by a distribution over choices (characterizing a type of person at a "stage of life"). As time proceeds, each person moves along layers of a tree branch, making choices based on the node at a given layer, thereby yielding a hierarchical representation of behavior with time. Note that as one moves deeper in the tree, the number of nodes at a given tree layer increases as a result of sequential branching; this appears to be well matched to the modeling of choices made by individuals, who often become more distinctive and specialized with increasing time. The number of paths (types of people), nodes (stages of development) and the statistics of the time dependence are to be inferred nonparametrically based on observed data, which are typically characterized by a very sparse binary matrix (most individuals only select a tiny fraction of the choices that are available to them).

To demonstrate this concept using real data, we consider selections of classes made by undergraduate students at Duke University, with the goal of uncovering structure in the students and classes, as inferred by time-evolving student choices. For each student, the data presented to the model are a set of indices of selected classes (but not class names or subject matter), as well as the academic year in which each class was selected (*e.g.*, sophomore year). While the student majors and class names are not used by the model, they are known, and this information provides "truth" with which model-inferred structure may be assessed. This study therefore also provides the opportunity to examine the quality of the inferred hierarchical-tree structure in models of the type considered in

[1, 4, 5, 8, 7, 12, 13, 6, 21, 15, 22] (such structure is difficult to validate with documents, for which there is no "truth"). We seek to impose that as time progresses it is more probable that an individual's choices are based on nodes deeper in the tree, so that as one moves from the tree root to the leaves, we observe the evolution of choices as people age. Such temporal tree-structure could be meaningful by itself, e.g., in our particular case it will allow university administrators and faculty to examine if objectives in curriculum design are manifested in the actual usage/choices of students. Further, the results of such an analysis may be of interest to applicants at a given school (e.g., high school students), as the inferred structure concisely describes both the student body and the curriculum. Also the uncovered structure may be used to aid downstream applications [17, 2, 11].

The basic form of the nonparametric tree developed here is based on the the nested Chinese restaurant process (nCRP) topic model [4, 5]. However, to achieve the goals of the unique problem considered, we make the following new modeling contributions. We develop a new "change-point" stick-breaking process (cpSBP), which is a stick-breaking process that induces probabilities that stochastically increase to an unknown changepoint and then decrease. This construction is conceptually related to the "umbrella" placed on dose response curves [9]. In the proposed model each individual has a unique cpSBP, that evolves with time such that choices at later times are encouraged to be associated with deeper nodes in the tree. Time is a covariate, and within the change-point model a new product-of-gammas construction is developed, and coupled to the Poisson distribution. Another novel aspect of the proposed model concerns drawing the node-dependent topics from a Dirichlet process, sharing topics across the tree. This is motivated by the idea that different types of people (paths) may be characterized by similar choices at different nodes in the respective paths (*e.g.*, person Type A may make certain types of choices early in life, while person Type B may make similar choices later in life). Such sharing of topics allows the inference of relationships between choices different people make over time.

## 2 Model Formulation

### 2.1 Nested Chinese Restaurant Process

The nested Chinese restaurant process (nCRP) [4, 5] is a generative probabilistic model that defines a prior distribution over a tree-structured hierarchy with infinitely many paths. In an nCRP model of personal choice each individual picks a tree path by walking from the root node down the tree, from node to node. Specifically, when situated at a particular parent node, the child node $c_i$ individual $i$ chooses is modeled as a random variable that can be either an existing node or a new node: ($i$) the probability that $c_i$ is an existing child node $k$ is proportional to the number of persons who already chose node $k$ from the same parent, ($ii$) and a new node can be created and chosen with probability proportional to $\gamma_0 > 0$, which is the nCRP concentration parameter. This process is defined recursively such that each individual is allocated to one specific path of the tree hierarchy, through a sequence of probabilistic parent-child steps.

The tree hierarchy implied by the nCRP provides a natural framework to capture the structure of personal choices, where each node is characterized by a distribution on the items that may be selected (*e.g.,* each node is a "choice topic"). Similar constructions have been considered for document analysis [5, 21, 4], in which the model captures the structure of word usage in documents. However, there are unique aspects of the time-evolving personal-choice problem, particularly the goal motivated above that one should select topics deeper in the tree as time evolves, to capture the specialized characteristics of people as they age. Hierarchical topic models proposed previously [4, 7] have employed a stick-breaking process (SBP) to guide selection of the tree depth at which a node/topic is selected, with an unbounded number of path layers, but these models do not provide a means of imposing the above temporal dynamics (which were not relevant for the document problems considered there).

### 2.2 Change Point Stick Breaking Process

In the proposed model, instead of constraining the SBP construction to start at the root node, we model the starting-point depth of the SBP as a random variable and infer it from data, while still maintaining a valid distribution over each layer of any path. To do this we replace the single SBP over path layers with *two* statistically dependent SBPs: one begins from layer $p+1$ and moves down

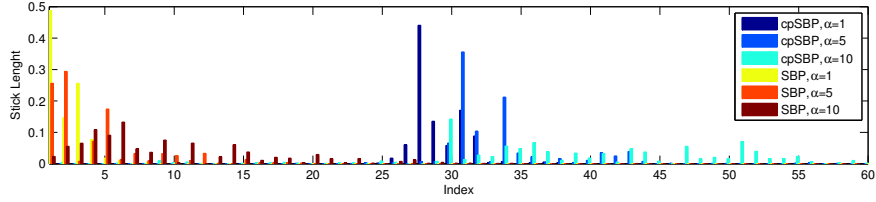

Figure 1: Illustrative comparison of the stick lengths between change point stick breaking process (cpSBP) and stick breaking process (SBP) with different value of $\alpha$; typical draws from cpSBP and SBP are depicted. $a_\omega$ and $b_\omega$ are both set to 1, the change point is set to $p = 30$ and the truncation of both stick breaking constructions are set to 60.

the tree away from the root, and the other begins from layer $p$ and moves upward to the root; the latter SBP is truncated when it hits the root, while the former is in principle of infinite length. The tree depth $p$ which relates these two SBPs is modeled as a random variable, drawn from a Poisson distribution, and is denoted the change point. In this way we encourage the principal stick weight to be placed heavily around the change point $p$, instead of restricting it to the top layers as in [4, 7]. To model the time dependence, and encourage use of greater tree depths with increasing time, we seek a formulation that imposes that the Poisson parameter grows (statistically) with increasing time.

The temporal information is represented as covariate $t(i, n)$, denoting the time at which the the $n$th selection/choice is made by individual $i$; in many applications $t(i, n) \in \{1, \ldots, T\}$, and for the student class-selection problem $T = 4$, corresponding to the freshman through senior years; below we drop the indices $(i, n)$ on the time, for notational simplicity. When individual $i$ makes selections at time $t$, she employs a corresponding change point $p_{i,t}$. To integrate the temporal covariate into the model, we develop a product-of-gammas and Poission conjugate pair to model $p_{i,t}$ which encourages $p_{i,t}$ associated with larger $t$ to locate deeper in the tree. Specifically, consider

$$\gamma_{i,l} = \text{Ga}(\gamma_{i,l}|a_{i,l}, b_{i,l}), \quad \lambda_{i,t} = \prod_{l=1}^{t} \gamma_{i,l}, \quad p_{i,t} \sim \text{Poi}(p_{i,t}|\lambda_{i,t}) \tag{1}$$

The product-of-gammas construction in (1) is inspired by the multiplicative-gamma process (MGP) developed in [3] for sparse factor analysis. Although each draw of $\gamma_{i,l}$ from a gamma distribution is not guaranteed to be greater than one, and thus $\lambda_{i,t}$ will not increase with probability one, in practice we find $\gamma_{i,l}$ is often inferred to be greater than one when $(a_{i,l} - 1)/b_{i,l} > 1$. However, an MGP based on a left-truncated gamma distribution can be readily derived.

Given the change point $p_{i,t} = p$, the cpSBP constructs the stick-weight vector $\theta_{i,t}^p$ over layers of path $b_i$ by dividing it to into two parts: $\hat{\theta}_{i,t}^p$ and $\tilde{\theta}_{i,t}^p$, modeling them separately as two SBPs, where $\hat{\theta}_{i,t}^p = \{\hat{\theta}_{i,t}^p(p), \hat{\theta}_{i,t}^p(p - 1), \ldots, \hat{\theta}_{i,t}^p(1)\}$ and $\tilde{\theta}_{i,t}^p = \{\tilde{\theta}_{i,t}^p(p + 1), \tilde{\theta}_{i,t}^p(p + 2), \ldots, \tilde{\theta}_{i,t}^p(\infty)\}$. For notation simplicity, we denote $V_h = V_{i,t}(h)$ when constructing $\theta_{i,t}^p$, yielding

$$\hat{\theta}_{i,t}^p(u) = V_u \prod_{h=u+1}^{p} (1 - V_h), \quad \tilde{\theta}_{i,t}^p(d) = V_d \prod_{h=p+1}^{d-1} (1 - V_h), \quad V_h \sim \text{beta}(V_h|1, \alpha) \tag{2}$$

Note that the above SBP contains two constructions: When $d > p$ the stick weight $\tilde{\theta}_{i,t}^p(d)$ is constructed as a classic SBP but with the stick-breaking construction starting at layer $p + 1$. On the other hand when $u \leq p$ the stick weight $\hat{\theta}_{i,t}^p(u)$ is constructed "backwards" from $p$ to the root node, which is a *truncated* stick breaking process with truncation level set to $p$. A thorough discussion of the truncated stick breaking process is found in [10]. We further use a beta distributed latent variable $\omega_{b_i}$ to combine the two stick breaking process together while ensuring each element of $\theta_{i,t}^p = \{\hat{\theta}_{i,t}^p, \tilde{\theta}_{i,t}^p\}$ sums to one. Thus we have the following distribution over layers of a given path from which the layer allocation variables $\{l_{i,n} : t(i, n) = t\}$ for a selection at time $t$ by individual $i$ are sampled:

$$l_{i,n} \sim \omega_{i,t} \sum_{l=1}^{p} \hat{\theta}_{i,t}(l)\delta_l + (1 - \omega_{i,t}) \sum_{l=p+1}^{\infty} \tilde{\theta}_{i,t}(l)\delta_l, \quad \omega_{i,t} \sim \text{Beta}(\omega_{i,t}|a_\omega, b_\omega) \tag{3}$$

Note that the change point stick breaking process (cpSBP) can be treated as a generalization of the stick breaking process for Dirichlet process, since when $p_{i,t} = 0$ the cpSBP corresponds to the SBP. From the simulation studied in Figure 1, we observe that the change point, which is modeled through the temporal covariate $t$ as in (1), corresponds to the layer with large stick weight and thus at which topic draws are most probable. Also note that one may alternatively suggest simply using $p_{i,t}$ directly as the layer from which a topic is drawn, without the subsequent use of a cpSBP. We examined this in the course of the experiments, and it did not work well, likely as a result of the inflexibility of the single-parameter Poisson (with its equal mean and variance). The cpSBP provided the additional necessary modeling flexibility.

### 2.3 Sharing topics among different nodes

One problem with the nCRP-based topic model, implied by the tree structure, is that all descendent sub-topics from parent node $pa_1$ are distinct from the descendants of parent $pa_2$, if $pa_1 \neq pa_2$. Some of these distinct sets of children from different parents may be redundant, and this redundancy can be removed if a child can have more than one parent [7, 13, 6]. In addition to the above problem, in our application there are other potential problems brought by the cpSBP. Since we encourage the later choice selections to be drawn from topics deeper in the tree, redundant topics at multiple layers may be manifested if two types of people tend to make similar choices at different time points (*e.g.,* at different stages of life). Thus it is likely that similar (redundant) topics may be learned on different layers of the tree, and the inability of the original nCRP construction to share these topics misses another opportunity to share statistical strength.

In [7, 13, 6] the authors addressed related challenges by replacing the tree structure with a directed acyclic graph (DAG), demonstrating success for document modeling. However, those solutions don't have the flexibility of sharing topics on nodes among different layers. Here we propose a new and simpler approach, so that the nCRP-based tree hierarchy is retained, while different nodes in the whole tree may share the same topic, resolving the two problems discussed above. To achieve this we draw a set of "global" topics $\{\hat{\phi}_k\}$, and a stick-breaking process is employed to allocate one of these global topics as $\phi_j$, representing the $j$th node in the tree (this corresponds to drawing the $\{\phi_j\}$ from a Dirichlet process [16], with a Dirichlet distribution base). The SBP defined over the global topics is represented as follows:

$$\pi_k = \delta_k \prod_{i=1}^{k-1} (1 - \delta_i), \quad \delta_i \sim \text{Beta}(\delta_i | 1, \eta), \quad \hat{\phi}_k \sim \text{Dir}(\hat{\phi}_k | \boldsymbol{\beta}), \quad \phi_j \sim \sum_{k=1}^{\infty} \pi_k \delta_{\hat{\phi}_k} \quad (4)$$

Within the generative process, let $z_{i,n}$ denote the assignment of the $n$th choice of individual $i$ to global topic $\hat{\phi}_{z_{i,n}}$; then the corresponding item chosen is drawn from $\text{Mult}(1, \hat{\phi}_{z_{i,n}})$.

## 3 Model Inference

In the proposed model, we sample the per-individual tree path indicator $b_i$, the layer allocation of choice topics in those paths $l_{i,n}$, the change point $p_{i,t}$ for each time interval, the parameters associated with the cpSBP construction $\gamma_{i,t}, \omega_{i,t}, \theta_{i,t}^p$, the stick breaking weight $\boldsymbol{\pi}$ over the global topics $\hat{\phi}_k$, and the global topic-assignment indicator $z_{i,n}$. Similar to [4], the per-node topic parameters $\phi_n$ are marginalized out. We provide update equations cycling through $\{l_{i,n}, p_{i,t}, \gamma_{i,t}, \omega_{i,t}, \boldsymbol{\theta}_{i,t}^p\}$ that are unique for this model. The update equations for $b_i$ and $\{\boldsymbol{\pi}, z_{i,n}\}$ are similar to the the ones in [4] and [18], respectively, which we do not reproduce here for brevity.

**Sampling for change point $p_{i,t}$** Due to the non-conjugacy between the Poisson and multinomial distributions, the exact form of its posterior distribution is difficult to compute. Additionally, in order to sample $p_{i,t}$, we require imputation of an infinite-dimensional process. The implementation of the sampling algorithm either relies on finite approximations [10] which lead to straightforward update equations, or requires an additional Metropolis-Hastings (M-H) step which allows us to obtain samples from the exact posterior distribution of $p_{i,t}$ with no approximation, *e.g.,* the retrospective sampler [14] proposed for Dirichlet process hierarchical models. In this section we first introduce the finite approximation based sampler, and the retrospective sampling scheme based method will be described in the supplemental material.

Denote $P$ as the truncated maximum value of the change point, then given the samples of all other latent variables, $p_{i,t}$ can be sampled from the following equation:

$$q(p_{i,t} = p|\boldsymbol{\theta}_{i,t}^p, \lambda_{i,t}, \omega_{i,t}, \boldsymbol{l}_{i,t}) \propto p(p_{i,t} = p|\lambda_{i,t}, P)p(\boldsymbol{l}_{i,t}|\boldsymbol{\theta}_{i,t}^p, \omega_{i,t}), \ 0 \le p \le P \qquad (5)$$

where $\boldsymbol{l}_{i,t} = \{l_{i,n} : t(i,n) = t\}$ are all layer allocations of choices made by individual $i$ at time $t$. $p(p_{i,t} = p|\lambda_{i,t}, P) = \frac{\lambda_{i,t}^p e^{-\lambda_{i,t}}}{p!C_P}$ is the Poisson density function truncated with $p \le P$, and $C_P = \sum_{p=1}^P \frac{\lambda_{i,t}^p e^{-\lambda_{i,t}}}{p!}$. $p(\boldsymbol{l}_{i,t}|\boldsymbol{\theta}_{i,t}^p, \omega_{i,t}) = \text{Mult}(\boldsymbol{l}_{i,t}|\{\omega_{i,t}\hat{\boldsymbol{\theta}}_{i,t}^p, (1 - \omega_{i,t})\tilde{\boldsymbol{\theta}}_{i,t}^p\})$ is the multinomial density function over the layer allocations $\boldsymbol{l}_{i,t}$.

**Sampling choice layer allocation** $l_{i,n}$  Given all the other variables, now we sample the layer allocation $l_{i,n}$ for the $n$th choice made by individual $i$. Denote $c_{i,n}$ as the $n$th choice made by individual $i$, $M_{z_{i,n},c_{i,n}} = \#[\boldsymbol{z}_{-(i,n)} = z_{i,n}, \boldsymbol{c}_{-(i,n)} = c_{i,n}] + \beta$ as the smoothed count of seeing choice $c_{i,n}$ allocated to global topic $z_{i,n}$, excluding the current choice. Parameter $l_{i,n}$ can be sampled from the following equation:

$$p(l_{i,n} = l|p_{i,t} = p, \boldsymbol{z}, \omega_{i,t}, \boldsymbol{\theta}_{i,t}^p, \boldsymbol{c}) \propto \begin{cases} \omega_{i,t}\hat{\theta}_{i,t}^p(l)M_{z_{i,n},c_{i,n}}, & 0 < l \le p \\ (1 - \omega_{i,t})\tilde{\theta}_{i,t}^p(l)M_{z_{i,n},c_{i,n}}, & p < l \le P \end{cases}$$

**Sampling for product-of-gammas construction** $\gamma_{i,t}$  From (1) note that the temporal dependent intensity parameter $\lambda_{i,t}$ can be reconstructed from the gamma distributed variables $\gamma_{i,t}$, which again can be sampled directly from its posterior distribution given all other variables, due to the conjugacy of product-gamma variable and Poisson construction. Denoting $\tau_{i,l}^{(t)} = \prod_{j=1, j\ne t}^l \gamma_{i,j}$, we have:

$$p(\gamma_{i,t}|\{p_{i,t}\}_{t=1}^T, a_{i,t}, b_{i,t}) = \text{Ga}\left(\gamma_{i,t}\Big| a_{i,t} + \sum_{l=t}^T p_{i,l}, b_{i,t} + \sum_{l=t}^T \tau_{i,l}^{(t)}\right)$$

**Sampling for cpSBP parameters** $\{\omega_{i,t}, \boldsymbol{\theta}_{i,t}^p\}$  Given the change points $p_{i,t}$ and choice layer allocation $\boldsymbol{l}_{i,t} = \{l_{i,n} : t(i,n) = t\}$, the cpSBP parameters $\boldsymbol{\theta}_{i,t}^p = \{\hat{\boldsymbol{\theta}}_{i,t}^p, \tilde{\boldsymbol{\theta}}_{i,t}^p\}$ can be reconstructed based on samples of $V_h$ as defined in (2). Specifically, we have

$$p(V_h|p_{i,t} = p, \boldsymbol{l}_{i,t}) = \begin{cases} \text{Beta}\left(V_h|a_h + N_{h,t}, b_h + \sum_{l=1}^{h-1} N_{l,t}\right) & \text{if } h \le p \\ \text{Beta}\left(V_h|a_h + N_{h,t}, b_h + \sum_{l=h+1}^{\max \boldsymbol{l}_{i,t}} N_{l,t}\right) & \text{if } h > p \end{cases}$$

where $N_{l,t} = \#[l_{i,n} = l, t(i,n) = t]$ records the number of times a choice made by individual $i$ in time interval $t$ is allocated to path layer $l$. Given the samples of other variables, $\omega_{i,t}$ is sampled from its full conditional posteior distribution: $p(\omega_{i,t}|p_{i,t} = p, \{l_{i,n} : t(i,n) = t\}) = \text{Beta}\left(\omega_{i,t}\Big|1 + \sum_{l=1}^p N_{l,t}, 1 + \sum_{l=p+1}^{\max \boldsymbol{l}_{i,t}} N_{l,t}\right)$

**Sampling the Hyperparameters**  Concerning hyperparameters $\alpha, \eta, \gamma_0, \boldsymbol{\beta}$, related to the stick breaking process and hierarchical topic model construction, we sample them within the inference process by placing prior distributions over them, similar to methods in [4]. One may also consider other alternatives for learning the hyperparameters within topic models [19]. For the hyperparameters $a_{i,l}, b_{i,l}$ in the product-of-gamma construction, we sample them as proposed in [3]. Finally, we fix $a_w = 1$ and sample $b_w$ by placing a gamma prior distribution on it. All these steps are done by a M-H step between iterations of the Gibbs sampler.

## 4   Analysis of student course selection

### 4.1   Data description and computations

We demonstrate the proposed model on real data by considering selections of classes made by undergraduate students at Duke University, for students in graduating classes 2009 to 2013; the data consists of class selections of all students from Fall 2005 to Spring 2011. For computational reasons

the cpSBP and SBP employed over the tree-path depth are truncated to a maximum of 10 layers (beyond this depth the number of topics employed by the data was minimal), while the number children of each parent node is allowed to be unbounded. Within the sampler, we ran the model based on the class selection records of students from class of 2009 and 2010 (total of 3382 students and 2756 unique classes), and collected 200 samples after burn-in, taking every fifth sample to approximate the posterior distribution over the latent tree structure as well as the topic on each node of the tree. We analyze the quality of the learned models using the remaining data (classes of 2011-2013), characterized by 5171 students and 2972 unique classes. Each topic is a probabilistic vector defined over 3015 classes offered across all years. Within the MCMC inference procedure we trained our model as follows: first, we fixed the change point $p_{i,t} = t$ and then ran the sampler for 100 iterations, then burned in the inference for 5000 iterations with $p_{i,t}$ updated before drawing 5000 samples from the full posterior.

## 4.2 Quantitative assessment

|  | # Topics | # Nodes | Predictive LL (11) | Predictive LL (11-13) |
|---|---|---|---|---|
| nCRP | 492±11 | 492±11 | -293226.8399 | -471736.8876 |
| cpSBP-nCRP | 973±37 | 973±37 | -290271.3576 | -469912.1120 |
| DP-nCRP | 318±26 | 521±41 | -292311.3971 | -471951.3452 |
| Full model | 367±32 | 961±44 | **-288511.4298** | **-468331.2990** |

Table 1: Predictive log-likelihood comparison on two datasets, given the mean of number of topics and nodes learned with rounded standard deviation. nCRP is the model proposed in [4]. Compared to nCRP, the cpSBP-nCRP replaced SBP with the proposed cpSBP, while in DP-nCRP the draw of topic for each node is from Dirichlet process(DP) instead of Dirichlet distribution and retained the SBP construction in nCRP. The full model used both cpSBP and DP. Results shown for class of 2011, and classes 2011-2013.

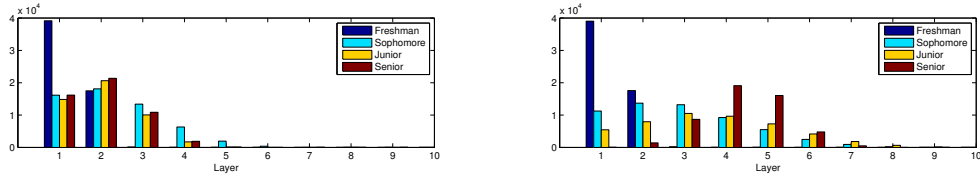

Figure 2: Histograms of class layer allocations according to their time covariates. Left: Stick breaking process, Right: Change point stick breaking process

In this section we examine the model's ability to explain unseen data. For comparison consistency we computed the predictive log-likelihood based on the samples collected in the same way as [4] (alternatives means of evaluating topic models are discussed in [20]). We test the model using two different compositions of the data, the first is based on class selection history of students from class of 2011 (1696 students), where all 4 years of records are available. The second is based on class selection history of students from class of 2011 to 2013 (3475 students), where for the later two years only partial course selection information is available, *e.g.,* for students from class of 2013 only class selection choices made in freshman year are available. Additionally, we compare the different models with respect to the learned number of topics and the learned number of tree nodes. This comparison is an indicator of the level of "parsimony" of the proposed model, introduced by replacing independent draws of topics from a Dirichlet distribution by draws from a Dirichlet process (with Dirichlet distribution base), as explained in Section 2.3. Since the number of tree nodes grows exponentially with the number of tree layers, from a practical viewpoint sharing topics among the nodes saves memory used to store the topic vectors, whose dimension is typically large (here the number of classes: 3015). In addition to the above insight, as the experimental results indicates, sharing topics among different nodes can enhance the sharing of statistical strength, which leads to better predictive performance. The results are summarized in Table 4.2.

We hypothesize that the enhanced performance of the proposed model to explain the unseen data is also due to it's improved ability to capture the latent predictive statistical structure, *e.g.,* to capture

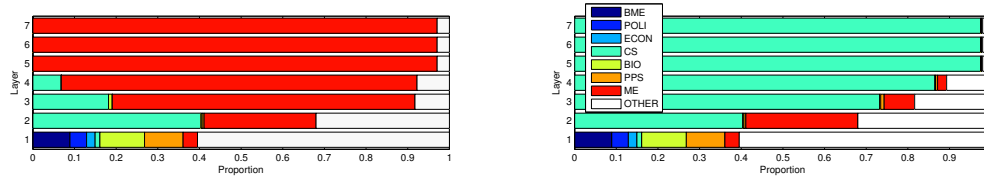

Figure 3: Change of the proportion of important majors along the layers of two paths which share the nodes up to the second layer. These two paths correspond to the full versions (all 7 layers) of the top two paths in Figure 4. BME: Biomedical Engineering, POLI: Political Science, ECON: Economics, CS: Computer Science, BIO: Biology, PPS: Public Policy Science, ME: Mechanical Engineering, OTHER: other 73 majors.

the latent temporal dynamics within the data by the change point stick breaking process (cpSBP). To demonstrate this point, in Figure 2 we compare how cpSBP and SBP guided the class layer allocations which have associated time covariates (*e.g.,* the academic year of each student). From Figure 2 we observe that under spSBP, as the students' academic career advances, they are more probable to choose classes from topics deeper in the tree, while such pattern is less obvious in the SBP case. Further, spSBP encouraged the data to utilize more layers in the tree than SBP.

### 4.3 Analyzing the learned tree

With incorporation of time covariates, we examine if the uncovered hierarchical structure is consistent with the actual curriculum of students from their freshman to senior year. And we consider two analyses here.

The first is a visualization of a subtree learned from the class-selection history, based on students of the class of 2009, as shown in Figure 4; shown are the most-probable classes in each topic, as well as a histogram on the covariates (1 to 4, for freshman through senior) of the students who employed the topic. For example, the topics on the top two layers correspond to the most popular classes selected by mechanical engineering and computer science students, respectively, while topics located to the right correspond to more advanced classes; to the left-most the root topic corresponds to classes required for all students (*e.g.,* academic writing). The tree structured hierarchy captured the general trend of class selection within/across different majors.

In Figure 4 we also highlight a topic in red, shared by two nodes. This topic corresponds to a set of general introductory classes which are popular (high attendance) for two types of students: ($i$) young students who take these classes early for preparation of future advanced studies, and ($ii$) students who need to fill elective requirements later in their academic career ("ideally" of an easy/elementary nature, to not "distract" from required classes from the major). It is therefore deemed interesting that these same classes seem to be preferred by young and old students, for apparently very different reasons. Note that the sharing of topics between nodes of different layers is a unique aspect of this model, not possible in [7, 13, 6].

In the second analysis we examine how the majors of students are distributed in the learned tree; the ideal case would be that each tree path corresponds to an academic major, and the nodes shared by paths manifest sharing of topics between different but related majors. In Figure 3 we show the change of proportions of different majors among different layers of the top two paths in Figure 4 (this is a zoom-in of a much larger tree). For a clear illustration, we show the seven most popular majors for these paths as a function of time (out of a total of 80 majors), and the remaining 73 majors are group together. We observe that the students with mechanical engineering (ME) majors share the node on the second layer with students with a computer science (CS) major, and the layers deeper in the tree begin to be exclusive to students with CS and ME majors, respectively. This phenomenon corresponds to the process a student determining her major by choosing courses as she walks down tree path. This also matches the fact that in this university, students declare their major during the sophomore year.

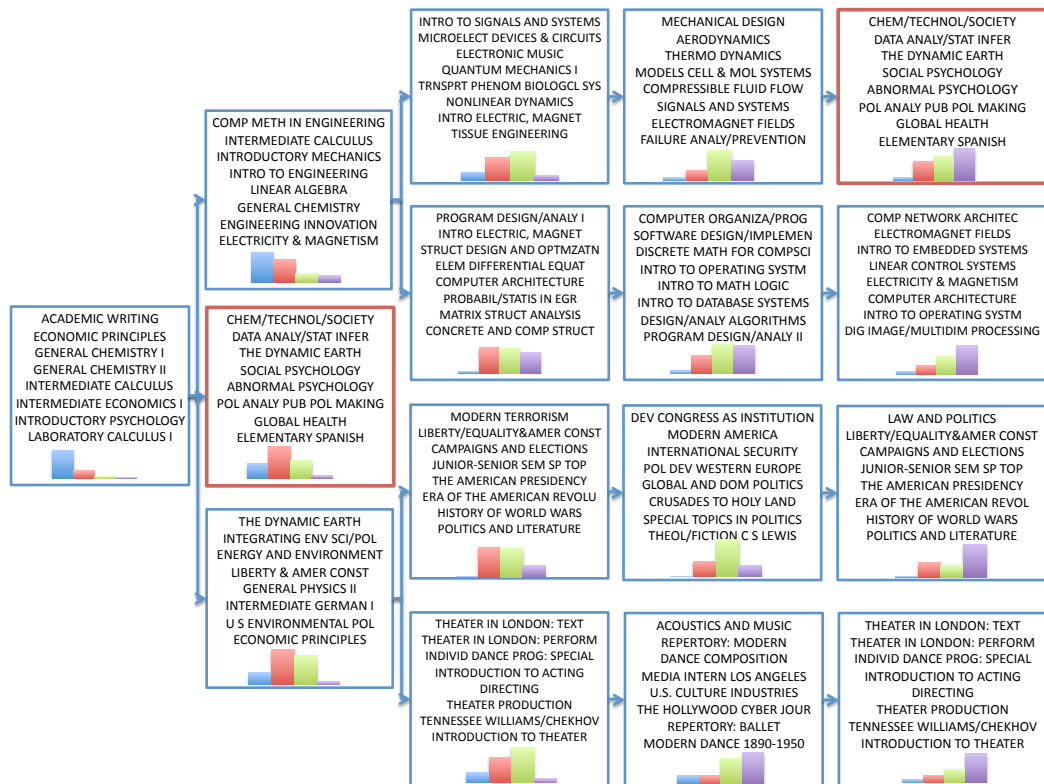

Figure 4: A subtree of topics learned from courses chosen by undergraduate students of class 2009; the whole tree has 372 nodes and 252 topics, and a maximum of 7 layers. Each node shows two aggregated statistics at that node: the eight most common classes of the topic on that node and a histogram of the academic year the topic was selected by students (1-4, for freshman-senior). The columns in each of the histogram correspond to freshman to senior year from left to right. The two highlighted red nodes share the same topic. These results correspond to one (maximum-likelihood) collection sample.

## 5 Discussion

We have extended hierarchical topic models to an important problem that has received limited attention to date: the evolution of personal choices over time. The proposed approach builds upon the nCRP [4], but introduces novel modeling components to address the problem of interest. Specifically, we develop a change-point stick-breaking process, coupled with a product of gammas and Poisson construction, that encourages individuals to be represented by nodes deeper in the tree as time evolves. The Dirichlet process has also been used to design the node-dependent topics, sharing strength and inferring relationships between choices of different people over time. The framework has been successfully demonstrated with a real-world data set: selection of courses over many years, for students at Duke University.

Although we worked only on one specific real-world data set, there are many other examples for which such a model may be of interest, especially when the data correspond to a sparse set of choices over time. For example, it could be useful for companies attempting to understand the purchases (choices) of customers, as a function of time (e.g., the clothing choices of people as they advance from teen years to adulthood). This may be of interest in marketing and targeted advertisement.

## Acknowledgements

We would like to thank the anonymous reviewers for their insightful comments. The research reported here was supported by AFOSR, ARO, DARPA, DOE, NGA and ONR.

# References

[1] R. P. Adams, Z. Ghahramani, and M. I. Jordan. Tree-structured stick breaking for hierarchical data. In *Neural Information Processing Systems (NIPS)*, 2010.

[2] E. Bart, I. Porteous, P. Perona, and M. Welling. Unsupervised learning of visual taxonomies. In *CVPR*, 2008.

[3] A. Bhattacharya and D. B. Dunson. Sparse Bayesian infinite factor models. *Biometrika*, 2011.

[4] D. M. Blei, T. L. Griffiths, and M. I. Jordan. The nested Chinese restaurant process and Bayesian nonparametric inference of topic hierarchies. *Journal of the ACM*, 57(2), 2010.

[5] D. M. Blei, T. L. Griffiths, M. I. Jordan, and J. B. Tenenbaum. Hierarchical topic models and the nested Chinese restaurant process. In *Neural Information Processing Systems (NIPS)*. 2004.

[6] A. Chambers, P. Smyth, and M. Steyvers. Learning concept graphs from text with stick-breaking priors. In *Advances in Neural Information Processing Systems(NIPS)*. 2010.

[7] H. Chen, D.B. Dunson, and L. Carin. Topic modeling with nonparametric Markov tree. In *Proc. Int. Conf. Machine Learning (ICML)*, 2011.

[8] T. L. Griffiths, M. Steyvers, and J. B. Tenenbaum. Topics in semantic representation. *Psychological Review*, 114(2):211–244, 2007.

[9] C. Hans and D. B. Dunson. Bayesian inferences on umbrella orderings. *BIOMETRICS*, 61:1018–1026, 2005.

[10] H. Ishwaran and L. F. James. Gibbs sampling methods for stick-breaking priors. *Journal of the American Statistical Association*, 96(453):161–173, 2001.

[11] L. Li, C. Wang, Y. Lim, D. Blei, and L. Fei-Fei. Building and using a semantivisual image hierarchy. In *CVPR*, 2010.

[12] W. Li and A. McCallum. Pachinko allocation: Dag-structured mixture models of topic correlations. In *Proc. Int. Conf. Machine Learning (ICML)*, 2006.

[13] D. Mimno, W. Li, and A. McCallum. Mixtures of hierarchical topics with Pachinko allocation. In *Proc. Int. Conf. Machine Learning (ICML)*, 2007.

[14] O. Papaspiliopoulos and G. O. Roberts. Retrospective Markov chain Monte Carlo methods for Dirichlet process hierarchiacal models. *Biometrika*, 95(1):169–186, 2008.

[15] R. Salakhutdinov, J. Tenenbaum, and A. Torralba. One-shot Learning with a Hierarchical Nonparametric Bayesian Model. *MIT Technical Report*, 2011.

[16] J. Sethuraman. A constructive definition of Dirichlet priors. *Statistica Sinica*, 4:639–650, 1994.

[17] J. Sivic, B. C. Russell, A. Zisserman, W. T. Freeman, and A. A. Efros. Unsupervised discovery of visual object class hierarchies. In *CVPR*, 2008.

[18] Y. W. Teh, M. I. Jordan, Matthew J. Beal, and D. M. Blei. Hierarchical Dirichlet processes. *Journal of the American Statistical Association*, 101(476):1566–1581, 2006.

[19] H. M. Wallach, D. Mimno, and A. McCallum. Rethinking LDA: Why priors matter. In *Neural Information Processing Systems (NIPS)*, 2009.

[20] H. M. Wallach, I. Murray, R. Salakhutdinov, and D. Mimno. Evaluation methods for topic models. In *Proc. Int. Conf. Machine Learning (ICML)*, 2009.

[21] C. Wang and D. M. Blei. Variational inference for the nested Chinese restaurant process. In *Neural Information Processing Systems (NIPS)*, 2009.

[22] XX. Zhang, D. B. Dunson, and L. Carin. Tree-structured infinite sparse factor model. In *Proc. Int. Conf. Machine Learning (ICML)*, 2011.

